# Hyperkernels

**Cheng Soon Ong, Alexander J. Smola, Robert C. Williamson**
Research School of Information Sciences and Engineering
The Australian National University
Canberra, 0200 ACT, Australia
{Cheng.Ong, Alex.Smola, Bob.Williamson}@anu.edu.au

## Abstract

We consider the problem of choosing a kernel suitable for estimation using a Gaussian Process estimator or a Support Vector Machine. A novel solution is presented which involves defining a Reproducing Kernel Hilbert Space on the space of kernels itself. By utilizing an analog of the classical represener theorem, the problem of choosing a kernel from a parameterized family of kernels (e.g. of varying width) is reduced to a statistical estimation problem akin to the problem of minimizing a regularized risk functional. Various classical settings for model or kernel selection are special cases of our framework.

## 1 Introduction

Choosing suitable kernel functions for estimation using Gaussian Processes and Support Vector Machines is an important step in the inference process. To date, there are few if any systematic techniques to assist in this choice. Even the restricted problem of choosing the "width" of a parameterized family of kernels (e.g. Gaussian) has not had a simple and elegant solution.

A recent development [1] which solves the above problem in a restricted sense involves the use of semidefinite programming to learn an arbitrary positive semidefinite matrix $K$, subject to minimization of criteria such as the kernel target alignment [1], the maximum of the posterior probability [2], the minimization of a learning-theoretical bound [3], or subject to cross-validation settings [4]. The restriction mentioned is that the methods work with the kernel matrix, rather than the kernel itself. Furthermore, whilst demonstrably improving the performance of estimators to some degree, they require clever parameterization and design to make the method work in the particular situations. There are still no general principles to guide the choice of a) which family of kernels to choose, b) efficient parameterizations over this space, and c) suitable penalty terms to combat overfitting. (The last point is particularly an issue when we have a very large set of semidefinite matrices at our disposal).

Whilst not yet providing a complete solution to these problems, this paper presents a framework that allows the optimization within a parameterized family relatively simply, and crucially, intrinsically captures the tradeoff between the size of the family of kernels and the sample size available. Furthermore, the solution presented *is* for optimizing kernels themselves, rather than the kernel matrix as in [1]. Other approaches on learning the kernel include using boosting [5] and by bounding the Rademacher complexity [6].

**Outline of the Paper** We show (Section 2) that for most kernel-based learning methods there exists a functional, the *quality functional*[1], which plays a similar role to the empirical risk functional, and that subsequently (Section 3) the introduction of a kernel on kernels, a so-called hyperkernel, in conjunction with regularization on the Reproducing Kernel Hilbert Space formed on kernels leads to a systematic way of parameterizing function classes whilst managing overfitting. We give several examples of hyperkernels (Section 4) and show (Section 5) how they can be used practically. Due to space constraints we only consider Support Vector classification.

## 2 Quality Functionals

Let $X_{\text{train}} := \{x_1, \ldots, x_m\}$ denote the set of training data and $Y_{\text{train}} := \{y_1, \ldots, y_m\}$ the set of corresponding labels, jointly drawn iid from some probability distribution $P(x, y)$ on $\mathcal{X} \times \mathcal{Y}$. Furthermore, let $X_{\text{test}}$ and $Y_{\text{test}}$ denote the corresponding test sets (drawn from the same $P(x, y)$). Let $X := X_{\text{train}} \cup X_{\text{test}}$ and $Y := Y_{\text{train}} \cup Y_{\text{test}}$.

We introduce a new class of functionals $Q$ on data which we call *quality functionals*. Their purpose is to indicate, given a kernel $k$ and the training data $(X_{\text{train}}, Y_{\text{train}})$, how suitable the kernel is for explaining the training data.

**Definition 1 (Empirical Quality Functional)** *Given a kernel $k$, and data $X, Y$, define $Q_{\text{emp}}[k, X, Y]$ to be an* empirical quality functional *if it depends on $k$ only via $k(x_i, x_j)$ where $x_i, x_j \in X$; i.e. if there exists a function $q$ such that $Q_{\text{emp}}[k, X, Y] = q(K, X, Y)$ where $K = [k(x_i, x_j)]_{i,j}$ is the* kernel matrix.

The basic idea is that $Q_{\text{emp}}$ could be used to adapt $k$ in a manner such that $Q_{\text{emp}}$ is minimized, based on this *single* dataset $X, Y$. Given a sufficiently rich class $\mathcal{K}$ of kernels $k$ it is in general possible to find a kernel $k^* \in \mathcal{K}$ that attains arbitrarily small values of $Q_{\text{emp}}[k^*, X_{\text{train}}, Y_{\text{train}}]$ for any training set. However, it is very unlikely that $Q_{\text{emp}}[k^*, X_{\text{test}}, Y_{\text{test}}]$ would be similarly small in general. Analogously to the standard methods of statistical learning theory, we aim to minimize the *expected* quality functional:

**Definition 2 (Expected Quality Functional)** *Suppose $Q_{\text{emp}}$ is an empirical quality functional. Then*
$$Q[k] := \mathbf{E}_{X,Y} \left[ Q_{\text{emp}}[k, X, Y] \right] \tag{1}$$
*is the expected quality functional, where the expectation is taken with respect to $P^m$.*

Note the similarity between $Q_{\text{emp}}[k, X, Y]$ and the empirical risk of an estimator $R_{\text{emp}}[f, X, Y] = \frac{1}{m} \sum_{i=1}^{m} c(x_i, y_i, f(x_i))$ (where $c$ is a suitable loss function): in both cases we compute the value of a functional which depends on some sample $X, Y$ drawn from $P(x, y)$ and a function, and in both cases we have

$$Q[k] = \mathbf{E}_{X,Y} \left[ Q_{\text{emp}}[k, X, Y] \right] \text{ and } R[f] = \mathbf{E}_{X,Y} \left[ R_{\text{emp}}[f, X, Y] \right]. \tag{2}$$

Here $R[f]$ is known as the expected risk. We now present some examples of quality functionals, and derive their exact minimizers whenever possible.

**Example 1 (Kernel Target Alignment)** *This quality functional was introduced in [7] to assess the "alignment" of a kernel with training labels. It is defined by*

$$Q_{\text{emp}}^{\text{alignment}}[k, X_{\text{train}}, Y_{\text{train}}] := 1 - \frac{y^\top K y}{\|y\|_2^2 \|K\|_2}, \tag{3}$$

*where $y$ denotes the vector of elements of $Y_{\text{train}}$, $\|y\|_2$ denotes the $\ell_2$ norm of $y$, and $\|K\|_2$ is the Frobenius norm: $\|K\|_2^2 := \text{tr} K K^\top = \sum_{i,j} K_{ij}^2$. Note that the definition in [7] looks somewhat different, yet it is algebraically identical to (3).*

By decomposing $K$ into its eigensystem, one can see that (3) is minimized if $K = yy^\top$, in which case

$$Q_{\text{emp}}^{\text{alignment}}[k^*, X_{\text{train}}, Y_{\text{train}}] = 1 - \frac{y^\top yy^\top y}{\|y\|_2^2 \|yy^\top\|_2} = 1 - \frac{\|y\|_2^4}{\|y\|_2^2 \|y\|_2^2} = 0. \qquad (4)$$

It is clear that one cannot expect that $Q_{\text{emp}}^{\text{alignment}}[k^*, X_{\text{train}}, Y_{\text{train}}] = 0$ for data other than the set chosen to determine $k^*$.

**Example 2 (Regularized Risk Functional)** *If $\mathcal{H}$ is the Reproducing Kernel Hilbert Space (RKHS) associated with the kernel $k$, the regularized risk functionals have the form*

$$R_{\text{reg}}[f, X_{\text{train}}, Y_{\text{train}}] := \frac{1}{m} \sum_{i=1}^m c(x_i, y_i, f(x_i)) + \frac{\lambda}{2} \|f\|_{\mathcal{H}}^2, \qquad (5)$$

*where $\|f\|_{\mathcal{H}}^2$ is the RKHS norm of $f$. By virtue of the representer theorem (see e.g., [4, 8]) we know that the minimizer over $f \in \mathcal{H}$ of (5) can be written as a kernel expansion. For a given loss $c$ this leads to the quality functional*

$$Q_{\text{emp}}^{\text{regrisk}}[k, X_{\text{train}}, Y_{\text{train}}] := \min_{\alpha \in \mathbb{R}^m} \left[ \frac{1}{m} \sum_{i=1}^m c(x_i, y_i, [K\alpha]_i) + \frac{\lambda}{2} \alpha^\top K \alpha \right]. \qquad (6)$$

*The minimizer of (6) is more difficult to find, since we have to carry out a double minimization over $K$ and $\alpha$. First, note that for $K = \beta yy^\top$ and $\alpha = \frac{1}{\beta \|y\|^2} y$, $K\alpha = y$ and $\alpha^\top K \alpha = \beta^{-1}$. Thus $Q_{\text{emp}}^{\text{regrisk}}[k, X_{\text{train}}, Y_{\text{train}}] = \frac{\lambda}{2\beta}$. For sufficiently large $\beta$, we can make $Q_{\text{emp}}^{\text{regrisk}}[k, X_{\text{train}}, Y_{\text{train}}]$ arbitrarily close to 0.*
*Even if we disallow setting $K$ to zero, by setting $\text{tr} K = 1$, we can determine the minimum of (6) as follows. Set $K = \frac{1}{\|z\|^2} zz^\top$, where $z \in \mathbb{R}^m$, and $\alpha = z$. Then $K\alpha = z$ and so*

$$\frac{1}{m} \sum_{i=1}^m c(x_i, y_i, [K\alpha]_i) + \frac{\lambda}{2} \alpha^\top K \alpha = \sum_{i=1}^m c(x_i, y_i, z_i) + \frac{\lambda}{2} \|z\|_2^2.$$

*Choosing each $z_i = \text{argmin}_\zeta c(x_i, y_i, \zeta) + \frac{\lambda}{2} \zeta^2$ yields the minimum with respect to $z$. The proof that $K$ is the* global *minimizer of this quality functional is omitted for brevity.*

**Example 3 (Negative Log-Posterior)** *In Gaussian processes, this functional is similar to $R_{\text{reg}}[f, X_{\text{train}}, Y_{\text{train}}]$ since it includes a regularization term (the negative log prior) and a loss term (the negative log-likelihood). In addition, it also includes the log-determinant of $K$ which measures the size of the space spanned by $K$. The quality functional is*

$$Q_{\text{emp}}^{\text{logpost}}[k, X_{\text{train}}, Y_{\text{train}}] := \min_{f \in \mathbb{R}^m} \left[ - \sum_{i=1}^m \log p(y_i | x_i, f_i) + \frac{1}{2} f^\top K^{-1} f + \frac{1}{2} \log |K| \right]$$
$$(7)$$

*Note that any $K$ which does not have full rank will send (7) to $-\infty$, and thus such cases need to be excluded. When we fix $|K| = 1$, to exclude the above case, we can set*

$$K = \beta \|y\|^{-2} yy^\top + \beta^{-\frac{1}{m-1}} (\mathbf{1} - \|y\|^{-2} yy^\top) \qquad (8)$$

*which leads to $|K| = 1$. Under the assumption that the minimum of $- \log p(y_i, x_i, f_i)$ with respect to $f_i$ is attained at $f_i = y_i$, we can see that $\beta \to \infty$ still leads to the overall minimum of $Q_{\text{emp}}^{\text{logpost}}[k, X_{\text{train}}, Y_{\text{train}}]$.*

Other examples, such as cross-validation, leave-one-out estimators, the Luckiness framework, the Radius-Margin bound also have empirical quality functionals which can be arbitrarily minimized.

The above examples illustrate how many existing methods for assessing the quality of a kernel fit within the quality functional framework. We also saw that given a rich enough class of kernels $\mathcal{K}$, optimization of $Q_{\text{emp}}$ over $\mathcal{K}$ would result in a kernel that would be useless for prediction purposes. This is yet another example of the danger of optimizing too much — there is (still) no free lunch.

# 3    A Hyper Reproducing Kernel Hilbert Space

We now introduce a method for optimizing quality functionals in an effective way. The method we propose involves the introduction of a Reproducing Kernel Hilbert Space *on the kernel $k$ itself* — a "Hyper"-RKHS. We begin with the basic properties of an RKHS (see Def 2.9 and Thm 4.2 in [8] and citations for more details).

**Definition 3 (Reproducing Kernel Hilbert Space)** *Let $\mathcal{X}$ be a nonempty set (often called the index set) and denote by $\mathcal{H}$ a Hilbert space of functions $f : \mathcal{X} \to \mathbb{R}$. Then $\mathcal{H}$ is called a reproducing kernel Hilbert space endowed with the dot product $\langle \cdot, \cdot \rangle$ (and the norm $\|f\| := \sqrt{\langle f, f \rangle}$) if there exists a function $k : \mathcal{X} \times \mathcal{X} \to \mathbb{R}$ satisfying, $x, x' \in \mathcal{X}$:*

1. *$k$ has the reproducing property $\langle f, k(x, \cdot) \rangle = f(x)$ for all $f \in \mathcal{H}$; in particular, $\langle k(x, \cdot), k(x', \cdot) \rangle = k(x, x')$.*
2. *$k$ spans $\mathcal{H}$, i.e. $\mathcal{H} = \overline{\text{span}\{k(x, \cdot) | x \in \mathcal{X}\}}$ where $\overline{X}$ is the completion of $X$.*

The advantage of optimization in an RKHS is that under certain conditions the optimal solutions can be found as the linear combination of a finite number of basis functions, regardless of the dimensionality of the space $\mathcal{H}$, as can be seen in the theorem below.

**Theorem 4 (Representer Theorem)** *Denote by $\Omega : [0, \infty) \to \mathbb{R}$ a strictly monotonic increasing function, by $\mathcal{X}$ a set, and by $c : (\mathcal{X} \times \mathbb{R}^2)^m \to \mathbb{R} \cup \{\infty\}$ an arbitrary loss function. Then each minimizer $f \in \mathcal{H}$ of the regularized risk*

$$c\left((x_1, y_1, f(x_1)), \ldots, (x_m, y_m, f(x_m))\right) + \Omega\left(\|f\|_{\mathcal{H}}\right) \tag{9}$$

*admits a representation of the form $f(x) = \sum_{i=1}^m \alpha_i k(x_i, x)$.*

The above definition allows us to define an RKHS on kernels $\mathcal{X} \times \mathcal{X} \to \mathbb{R}$, simply by introducing $\underline{\mathcal{X}} := \mathcal{X} \times \mathcal{X}$ and by treating $k$ as functions $k : \underline{\mathcal{X}} \to \mathbb{R}$:

**Definition 5 (Hyper Reproducing Kernel Hilbert Space)** *Let $\mathcal{X}$ be a nonempty set and let $\underline{\mathcal{X}} := \mathcal{X} \times \mathcal{X}$ (the compounded index set). Then the Hilbert space $\underline{\mathcal{H}}$ of functions $k : \underline{\mathcal{X}} \to \mathbb{R}$, endowed with a dot product $\langle \cdot, \cdot \rangle$ (and the norm $\|k\| = \sqrt{\langle k, k \rangle}$) is called a* Hyper Reproducing Kernel Hilbert Space *if there exists a* hyperkernel *$\underline{k} : \underline{\mathcal{X}} \times \underline{\mathcal{X}} \to \mathbb{R}$ with the following properties:*

1. *$\underline{k}$ has the reproducing property $\langle k, \underline{k}(\underline{x}, \cdot) \rangle = k(\underline{x})$ for all $k \in \underline{\mathcal{H}}$, in particular, $\langle \underline{k}(\underline{x}, \cdot), \underline{k}(\underline{x}', \cdot) \rangle = \underline{k}(\underline{x}, \underline{x}')$.*
2. *$\underline{k}$ spans $\underline{\mathcal{H}}$, i.e. $\underline{\mathcal{H}} = \overline{\text{span}\{\underline{k}(\underline{x}, \cdot) | \underline{x} \in \underline{\mathcal{X}}\}}$.*
3. *For any fixed $\underline{x} \in \underline{\mathcal{X}}$ the hyperkernel $\underline{k}$ is a kernel in its second argument, i.e. for any fixed $\underline{x} \in \underline{\mathcal{X}}$, the function $k(x, x') := \underline{k}(\underline{x}, (x, x'))$ with $x, x' \in \mathcal{X}$ is a kernel.*

What distinguishes $\underline{\mathcal{H}}$ from a normal RKHS is the particular form of its index set ($\underline{\mathcal{X}} = \mathcal{X}^2$) and the additional condition on $\underline{k}$ to be a kernel in its second argument for any fixed first argument. This condition somewhat limits the choice of possible kernels. On the other hand, it allows for simple optimization algorithms which consider kernels $k \in \underline{\mathcal{H}}$, which are in the convex cone of $\underline{k}$. Analogously to the definition of the regularized risk functional (5), we define the regularized quality functional:

$$Q_{\text{reg}}[k, X, Y] := Q_{\text{emp}}[k, X, Y] + \frac{\lambda_s}{2}\|k\|^2, \tag{10}$$

where $\lambda_s > 0$ is a regularization constant and $\|k\|^2$ denotes the RKHS norm in $\underline{\mathcal{H}}$. Minimization of $Q_{\text{reg}}$ is less prone to overfitting than minimizing $Q_{\text{emp}}$, since the regularization term $\frac{\lambda_s}{2}\|k\|^2$ effectively controls the complexity of the class of kernels under consideration. Regularizers other than $\frac{\lambda_s}{2}\|k\|^2$ are also possible. The question arising immediately from (10) is how to minimize the regularized quality functional efficiently. In the following we show that the minimum can be found as a linear combination of hyperkernels.

**Corollary 6 (Representer Theorem for Hyper-RKHS)** *Let $\underline{\mathcal{H}}$ be a hyper-RKHS and denote by $\Omega : [0, \infty) \to \mathbb{R}$ a strictly monotonic increasing function, by $\mathcal{X}$ a set, and by $Q$ an arbitrary quality functional. Then each minimizer $k \in \underline{\mathcal{H}}$ of the regularized quality functional*

$$Q[k, X, Y] + \frac{\lambda_s}{2}\|k\|^2 \tag{11}$$

*admits a representation of the form $k(x, x') = \sum_{i,j=1}^{m} \beta_{ij}\underline{k}((x_i, x_j), (x, x')).$*

**Proof** All we need to do is rewrite (11) so that it satisfies the conditions of Theorem 4. Let $\underline{x}_{ij} := (x_i, x_j)$. Then $Q[k, X, Y]$ has the properties of a loss function, as it only depends on $k$ via its values at $\underline{x}_{ij}$. Furthermore, $\frac{\lambda_s}{2}\|k\|^2$ is an RKHS regularizer, so the representer theorem applies and the expansion of $k$ follows. ∎

This result shows that even though we are optimizing over an entire (potentially infinite dimensional) Hilbert space of kernels, we are able to find the optimal solution by choosing among a finite dimensional subspace. The dimension required ($m^2$) is, not surprisingly, significantly larger than the number of kernels required in a kernel function expansion which makes a direct approach possible only for small problems. However, sparse expansion techniques, such as [9, 8], can be used to make the problem tractable in practice.

## 4 Examples of Hyperkernels

Having introduced the theoretical basis of the Hyper-RKHS, we need to answer the question whether practically useful $\underline{k}$ exist which satisfy the conditions of Definition 5. We address this question by giving a set of general recipes for building such kernels.

**Example 4 (Power Series Construction)** *Denote by $k$ a positive semidefinite kernel, and by $g : \mathbb{R} \to \mathbb{R}$ a function with positive Taylor expansion coefficients $g(\xi) = \sum_{i=0}^{\infty} c_i \xi^i$ and convergence radius $R$. Then for $k^2(x, x') \le R$ we have that*

$$\underline{k}(\underline{x}, \underline{x}') := g(k(\underline{x})k(\underline{x}')) = \sum_{i=0}^{\infty} c_i (k(\underline{x})k(\underline{x}'))^i \tag{12}$$

*is a hyperkernel: for any fixed $\underline{x}$, $\underline{k}(\underline{x}, (x, x'))$ is a sum of kernel functions, hence it is a kernel itself (since $k^i(x, x')$ is a kernel if $k$ is). To show that $\underline{k}$ is a kernel, note that $\underline{k}(\underline{x}, \underline{x}') = \langle \underline{\Phi}(\underline{x}), \underline{\Phi}(\underline{x}')\rangle$, where $\underline{\Phi}(\underline{x}) := (\sqrt{c_0}, \sqrt{c_1}k^1(\underline{x}), \sqrt{c_2}k^2(\underline{x}), \ldots).$*

**Example 5 (Harmonic Hyperkernel)** *A special case of (12) is the harmonic hyperkernel: Denote by $k$ a kernel with $k : \mathcal{X} \times \mathcal{X} \to [0, 1]$ (e.g., RBF kernels satisfy this property), and set $c_i := (1 - \lambda_h)\lambda_h^i$ for some $0 < \lambda_h < 1$. Then we have*

$$\underline{k}(\underline{x}, \underline{x}') = (1 - \lambda_h)\sum_{i=0}^{\infty} \left(\lambda_h k(\underline{x})k(\underline{x}')\right)^i = \frac{1 - \lambda_h}{1 - \lambda_h k(\underline{x})k(\underline{x}')}. \tag{13}$$

**Example 6 (Gaussian Harmonic Hyperkernel)** *For $k(x, x') = \exp(-\sigma^2\|x - x'\|^2)$,*

$$\underline{k}((x, x'), (x'', x''')) = \frac{1 - \lambda_h}{1 - \lambda_h \exp\left(-\sigma^2(\|x - x'\|^2 + \|x'' - x'''\|^2)\right)}. \tag{14}$$

*For $\lambda_h \to 1$, $\underline{k}$ converges to $\delta_{\underline{x},\underline{x}'}$; that is, the expression $\|k\|^2$ converges to the Frobenius norm of $k$ on $X \times X$.*

| $g(\xi)$ | Power series expansion | $R$ |
|---|---|---|
| $\exp \xi$ | $1 + \frac{1}{1!}\xi + \ldots + \frac{1}{n!}\xi^n + \ldots$ | $\infty$ |
| $\sinh \xi$ | $\frac{\xi}{1!} + \frac{\xi^3}{3!} + \ldots + \frac{\xi^{(2n+1)}}{(2n+1)!} + \ldots$ | $\infty$ |
| $\cosh \xi$ | $1 + \frac{\xi^2}{2!} + \ldots + \frac{\xi^{(2n)}}{(2n)!} + \ldots$ | $\infty$ |
| $\mathrm{arctanh}\,\xi$ | $\frac{\xi}{1} + \frac{\xi^3}{3} + \ldots + \frac{\xi^{2n+1}}{2n+1} + \ldots$ | $1$ |
| $-\ln(1-\xi)$ | $\frac{\xi}{1} + \frac{\xi^2}{2} + \ldots + \frac{\xi^n}{n} + \ldots$ | $1$ |

**Table 1:** Examples of Hyperkernels

We can find further hyperkernels, simply by consulting tables on power series of functions. Table 1 contains a list of suitable expansions. Recall that expansions such as (12) were mainly chosen for computational convenience, in particular whenever it is not clear which particular class of kernels would be useful for the expansion.

**Example 7 (Explicit Construction)** *If we* know *or have a reasonable guess as to which kernels could be potentially relevant (e.g., a range of scales of kernel width, polynomial degrees, etc.), we may begin with a set of candidate kernels, say $k_1, \ldots, k_n$ and define*

$$\underline{k}(\underline{x}, \underline{x}') := \sum_{i=1}^{n} c_i k_i(\underline{x}) k_i(\underline{x}'), \quad k_i(\underline{x}) \geqslant 0, \forall \underline{x}. \tag{15}$$

*Clearly $\underline{k}$ is a hyperkernel, since $\underline{k}(\underline{x}, \underline{x}') = \langle \underline{\Phi}(\underline{x}), \underline{\Phi}(\underline{x}') \rangle$, where $\underline{\Phi}(\underline{x}) := (\sqrt{c_1} k_1(\underline{x}), \sqrt{c_2} k_2(\underline{x}), \ldots, \sqrt{c_n} k_n(\underline{x}))$.*

## 5  An Application: Minimization of the Regularized Risk

Recall that in the case of the Regularized Risk functional, the regularized quality optimization problem takes on the form

$$\underset{f \in \mathcal{H}, k \in \underline{\mathcal{H}}}{\text{minimize}} \frac{1}{m} \sum_{i=1}^{m} c(x_i, y_i, f(x_i)) + \frac{\lambda}{2}\|f\|_{\mathcal{H}}^2 + \frac{\lambda_s}{2}\|k\|_{\underline{\mathcal{H}}}^2. \tag{16}$$

For $f = \sum_i \alpha_i k(x_i, x)$, the second term $\|f\|_{\mathcal{H}}^2$ is a linear function of $k$. Given a convex loss function $c$, the regularized quality functional (16) is convex in $k$. The corresponding regularized quality functional is:

$$Q_{\text{reg}}^{\text{regrisk}}[k, X, Y] = Q_{\text{emp}}^{\text{regrisk}}[k, X, Y] + \frac{\lambda_s}{2}\|k\|_{\underline{\mathcal{H}}}^2 \tag{17}$$

For fixed $k$, the problem can be formulated as a constrained minimization problem in $f$, and subsequently expressed in terms of the Lagrange multipliers $\alpha$. However, this minimum depends on $k$, and for efficient minimization we would like to compute the derivatives with respect to $k$. The following lemma tells us how (it is an extension of a result in [3] and we omit the proof for brevity):

**Lemma 7** *Let $x \in \mathbb{R}^m$ and denote by $f(x, \theta), c_i : \mathbb{R}^m \to \mathbb{R}$ convex functions, where $f$ is parameterized by $\theta$. Let $R(\theta)$ be the minimum of the following optimization problem (and denote by $x(\theta)$ its minimizer):*

$$\underset{x \in \mathbb{R}^m}{\text{minimize}} \ f(x, \theta) \text{ subject to } c_i(x) \leq 0 \text{ for all } 1 \leq i \leq n. \tag{18}$$

*Then $\partial_\theta^j R(\theta) = D_2^j f(x(\theta), \theta)$, where $j \in \mathbb{N}$ and $D_2$ denotes the derivative with respect to the second argument of $f$.*

Since the minimizer of (17) can be written as a kernel expansion (by the representer theorem for Hyper-RKHS), the optimal regularized quality functional can be written as (using

the soft margin loss and $\underline{K}_{ijpq} := \underline{k}((x_i, x_j), (x_p, x_q))$:

$$
\begin{aligned}
Q_{\text{reg}}^{\text{regrisk}}[\underline{K}, \alpha, \beta, X, Y] \quad = \quad & \frac{1}{m} \sum_{i=1}^{m} \max\left(0, 1 - y_i \sum_{j,p,q=1}^{m} \alpha_j \beta_{pq} \underline{K}_{ijpq}\right) \qquad (19) \\
+ \quad & \frac{\lambda}{2} \sum_{i,j,p,q=1}^{m} \alpha_i \alpha_j \beta_{pq} \underline{K}_{ijpq} + \frac{\lambda'}{2} \sum_{i,j,p,q=1}^{m} \beta_{ij} \beta_{pq} \underline{K}_{ijpq}
\end{aligned}
$$

Minimization of (19) is achieved by alternating between minimization over $\alpha$ for fixed $\beta$ (this is a quadratic optimization problem), and subsequently minimization over $\beta$ (with $\beta_{ij} \geq 0$ to ensure positivity of the kernel matrix) for fixed $\alpha$.

**Low Rank Approximation**   While being finite in the number of parameters (despite the optimization over two possibly infinite dimensional Hilbert spaces $\underline{\mathcal{H}}$ and $\mathcal{H}$), (19) still presents a formidable optimization problem in practice (we have $m^2$ coefficients for $\beta$). For an explicit expansion of type (15) we can optimize in the expansion coefficients of $k_i(\underline{x})k_i(\underline{x}')$ directly, which means that we simply have a quality functional with an $\ell_2$ penalty on the expansion coefficients. Such an approach is recommended if there are few terms in (15). In the general case (or if $n \gg m$), we resort to a low-rank approximation, as described in [9, 8]. This means that we pick from $\underline{k}((x_i, x_j), \cdot)$ with $1 \leq i, j \leq m$ a small fraction of terms which approximate $\underline{k}$ on $X \times X$ sufficiently well.

## 6   Experimental Results and Summary

**Experimental Setup**   To test our claims of kernel adaptation via regularized quality functionals we performed preliminary tests on datasets from the UCI repository (Pima, Ionosphere, Wisconsin diagnostic breast cancer) and the USPS database of handwritten digits ('6' vs. '9'). The datasets were split into 60% training data and 40% test data, except for the USPS data, where the provided split was used. The experiments were repeated over 200 random 60/40 splits. We deliberately did not attempt to tune parameters and instead made the following choices uniformly for all four sets:

- The kernel width $\sigma$ was set to $\sigma^{-1} = 100d$, where $d$ is the dimensionality of the data. We deliberately chose a too large value in comparison with the usual rules of thumb [8] to avoid good default kernels.
- $\lambda$ was adjusted so that $\frac{1}{\lambda m} = 100$ (that is $C = 100$ in the Vapnik-style parameterization of SVMs). This has commonly been reported to yield good results.
- $\lambda_h$ for the Gaussian Harmonic Hyperkernel was chosen to be 0.6 throughout, giving adequate coverage over various kernel widths in (13) (small $\lambda_h$ focus almost exclusively on wide kernels, $\lambda_h$ close to 1 will treat all widths equally).
- The hyperkernel regularization was set to $\lambda_s = 10^{-4}$.

We compared the results with the performance of a generic Support Vector Machine with the same values chosen for $\sigma$ and $\lambda$ and one for which $\lambda, \sigma$ had been hand-tuned using cross validation.

**Results**   Despite the fact that we did not try to tune the parameters we were able to achieve highly competitive results as shown in Table 2. It is also worth noticing that the number of hyperkernels required after a low-rank decomposition of the hyperkernel matrix contained typically less than 10 hyperkernels, thus rendering the optimization problem not much more costly than a standard Support Vector Machine (even with a very high quality $10^{-5}$ approximation of $\underline{K}$) and that after the optimization of (19), typically less than 5 were being used. This dramatically reduced the computational burden.

Using the *same* non-optimized parameters for different data sets we achieved results comparable to other recent work on classification such as boosting, optimized SVMs, and kernel target alignment [10, 11, 7] (note that we use a much smaller part of the data for training:

| Data(size) | $R_{\text{reg}}$ | | $Q_{\text{reg}}$ | | Best in [10, 11] | Tuned SVM |
|---|---|---|---|---|---|---|
| | Train | Test | Train | Test | | |
| pima(768) | 25.2±2.0 | 26.2±3.3 | 22.2±1.4 | 23.2±2.0 | 23.5 | 22.9±2.0 |
| ionosph(351) | 13.4±2.0 | 16.5±3.4 | 10.9±1.5 | 13.4±2.4 | 6.2 | 6.1±1.9 |
| wdbc(569) | 5.7±0.8 | 5.7±1.3 | 2.1±0.6 | 2.7±1.0 | 3.2 | 2.5±0.9 |
| usps(1424) | 2.1 | 3.4 | 1.5 | 2.8 | NA | 2.5 |

Table 2: Training and test error in percent

only 60% rather than 90%). Results based on $Q_{\text{reg}}$ are comparable to hand tuned SVMs (right most column), except for the ionosphere data. We suspect that this is due to the small training sample.

**Summary and Outlook** The regularized quality functional allows the systematic solution of problems associated with the choice of a kernel. Quality criteria that can be used include target alignment, regularized risk and the log posterior. The regularization implicit in our approach allows the control of overfitting that occurs if one optimizes over a too large a choice of kernels.

A very promising aspect of the current work is that it opens the way to theoretical analyses of the price one pays by optimizing over a larger set $\mathcal{K}$ of kernels. Current and future research is devoted to working through this analysis and subsequently developing methods for the design of good hyperkernels.

**Acknowledgements** This work was supported by a grant of the Australian Research Council. The authors thank Grace Wahba for helpful comments and suggestions.

## Footnotes

[1] We actually mean *badness*, since we are minimizing this functional.

# References

[1] G. Lanckriet, N. Cristianini, P. Bartlett, L. El Ghaoui, and M. Jordan. Learning the kernel matrix with semidefinite programming. In *ICML*. Morgan Kaufmann, 2002.

[2] C. K. I. Williams. Prediction with Gaussian processes: From linear regression to linear prediction and beyond. In M. I. Jordan, editor, *Learning and Inference in Graphical Models*. Kluwer Academic, 1998.

[3] O. Chapelle, V. Vapnik, O. Bousquet, and S. Mukherjee. Choosing kernel parameters for support vector machines. *Machine Learning*, 2002. Forthcoming.

[4] G. Wahba. *Spline Models for Observational Data*, volume 59 of *CBMS-NSF Regional Conference Series in Applied Mathematics*. SIAM, Philadelphia, 1990.

[5] K. Crammer, J. Keshet, and Y. Singer. Kernel design using boosting. In *Advances in Neural Information Processing Systems 15*, 2002. In press.

[6] O. Bousquet and D. Herrmann. On the complexity of learning the kernel matrix. In *Advances in Neural Information Processing Systems 15*, 2002. In press.

[7] N. Cristianini, A. Elisseeff, and J. Shawe-Taylor. On optimizing kernel alignment. Technical Report NC2-TR-2001-087, NeuroCOLT, http://www.neurocolt.com, 2001.

[8] B. Schölkopf and A. J. Smola. *Learning with Kernels*. MIT Press, 2002.

[9] S. Fine and K. Scheinberg. Efficient SVM training using low-rank kernel representation. Technical report, IBM Watson Research Center, New York, 2000.

[10] Y. Freund and R. E. Schapire. Experiments with a new boosting algorithm. In *ICML*, pages 148–146. Morgan Kaufmann Publishers, 1996.

[11] G. Rätsch, T. Onoda, and K. R. Müller. Soft margins for adaboost. *Machine Learning*, 42(3):287–320, 2001.